# An Extended Level Method for Efficient Multiple Kernel Learning

**Zenglin Xu**[†]    **Rong Jin**[‡]    **Irwin King**[†]    **Michael R. Lyu**[†]

[†] Dept. of Computer Science & Engineering
The Chinese University of Hong Kong
Shatin, N.T., Hong Kong
{zlxu, king, lyu}@cse.cuhk.edu.hk

[‡]Dept. of Computer Science & Engineering
Michigan State University
East Lansing, MI, 48824
rongjin@cse.msu.edu

## Abstract

We consider the problem of multiple kernel learning (MKL), which can be formulated as a convex-concave problem. In the past, two efficient methods, i.e., Semi-Infinite Linear Programming (SILP) and Subgradient Descent (SD), have been proposed for large-scale multiple kernel learning. Despite their success, both methods have their own shortcomings: (a) the SD method utilizes the gradient of only the current solution, and (b) the SILP method does not regularize the approximate solution obtained from the cutting plane model. In this work, we extend the level method, which was originally designed for optimizing non-smooth objective functions, to convex-concave optimization, and apply it to multiple kernel learning. The extended level method overcomes the drawbacks of SILP and SD by exploiting all the gradients computed in past iterations and by regularizing the solution via a projection to a level set. Empirical study with eight UCI datasets shows that the extended level method can significantly improve efficiency by saving on average $91.9\%$ of computational time over the SILP method and $70.3\%$ over the SD method.

## 1 Introduction

Kernel learning [5, 9, 7] has received a lot of attention in recent studies of machine learning. This is due to the importance of kernel methods in that kernel functions define a generalized similarity measure among data. A generic approach to learning a kernel function is known as multiple kernel learning (MKL) [5]: given a list of base kernel functions/matrices, MKL searches for the linear combination of base kernel functions which maximizes a generalized performance measure. Previous studies [5, 14, 13, 4, 1] have shown that MKL is usually able to identify appropriate combination of kernel functions, and as a result to improve the performance.

A variety of methods have been used to create base kernels. For instance, base kernels can be created by using different kernel functions; they can also be created by using a single kernel function but with different subsets of features. As for the performance measures needed to find the optimal kernel function, several measures have been studied for multiple kernel learning, including maximum margin classification errors [5], kernel-target alignment [4], and Fisher discriminative analysis [13].

The multiple kernel learning problem was first formulated as a semi-definite programming (SDP) problem by [5]. An SMO-like algorithm was proposed in [2] in order to solve medium-scale problems. More recently, a Semi-Infinite Linear Programming (SILP) approach was developed for MKL [12]. SILP is an iterative algorithm that alternates between the optimization of kernel weights and the optimization of the SVM classifier. In each step, given the current solution of kernel weights, it solves a classical SVM with the combined kernel; it then constructs a cutting plane model for the objective function and updates the kernel weights by solving a corresponding linear programming

problem. Although the SILP approach can be employed for large scale MKL problems, it often suffers from slow convergence. One shortcoming of the SILP method is that it updates kernel weights solely based on the cutting plane model. Given that a cutting plane model usually differs significantly from the original objective function when the solution is far away from the points where the cutting plane model is constructed, the optimal solution to the cutting plane model could be significantly off target. In [10], the authors addressed the MKL problems by a simple Subgradient Descent (SD) method. However, since the SD method is memoryless, it does not utilize the gradients computed in previous iterations, which could be very useful in boosting the efficiency of the search.

To further improve the computational efficiency of MKL, we extended the level method [6], which was originally designed for optimizing non-smooth functions, to the optimization of convex-concave problems. In particular, we regard the MKL problem as a saddle point problem. In the present work, similar to the SILP method, we construct in each iteration a cutting plane model for the target objective function using the solutions to the intermediate SVM problems. A new solution for kernel weights is obtained by solving the cutting plane model. We furthermore adjust the new solution via a projection to a level set. This adjustment is critical in that it ensures on one hand the new solution is sufficiently close to the current solution, and on the other hand the new solution significantly reduces the objective function. We show that the extended level method has a convergence rate of $\mathcal{O}(1/\varepsilon^2)$ for a $\varepsilon$-accurate solution. Although this is similar to that of the SD method, the extended level method is advantageous in that it utilizes all the gradients that have been computed so far. Empirical results with eight UCI datasets show that the extended level method is able to greatly improve the efficiency of multiple kernel learning in comparison with the SILP method and the SD method.

The rest of this paper is organized as follows. In section 2, we review the efficient algorithms that have been designed for multiple kernel learning. In section 3, we describe the details of the extended level method for MKL, including a study of its convergence rate. In section 4, we present experimental results by comparing both the effectiveness and the efficiency of the extended level method with the corresponding measures of SILP and SD. We conclude this work in section 5.

## 2   Related Work

Let $\mathbf{X} = (\mathbf{x}_1, \ldots, \mathbf{x}_n) \in \mathbb{R}^{n \times d}$ denote the collection of $n$ training samples that are in a $d$-dimensional space. We further denote by $\mathbf{y} = (y_1, y_2, \ldots, y_n) \in \{-1, +1\}^n$ the binary class labels for the data points in $\mathbf{X}$. We employ the maximum margin classification error, an objective used in SVM, as the generalized performance measure. Following [5], the problem of multiple kernel learning for classification in the primal form is defined as follows:

$$\min_{\mathbf{p} \in \mathcal{P}} \max_{\alpha \in \mathcal{Q}} \; f(\mathbf{p}, \alpha) = \alpha^\top \mathbf{e} - \frac{1}{2}(\alpha \circ \mathbf{y})^\top \left( \sum_{i=1}^{m} p_i \mathbf{K}_i \right) (\alpha \circ \mathbf{y}), \qquad (1)$$

where $\mathcal{P} = \{\mathbf{p} \in \mathbb{R}^m : \mathbf{p}^\top \mathbf{e} = 1, \; 0 \leq \mathbf{p} \leq 1\}$ and $\mathcal{Q} = \{\alpha \in \mathbb{R}^n : \alpha^\top \mathbf{y} = 0, \; 0 \leq \alpha \leq C\}$ are two solid convex regions, denoting the set of kernel weights and the set of SVM dual variables, respectively. Here, $\mathbf{e}$ is a vector of all ones, $C$ is the trade-off parameter in SVM, $\{\mathbf{K}_i\}_{i=1}^{m}$ is a group of base kernel matrices, and $\circ$ defines the element-wise product between two vectors. It is easy to verify that $f(\mathbf{p}, \alpha)$ is convex on $\mathbf{p}$ and concave on $\alpha$. Thus the above optimization problem is indeed a convex-concave problem. It is important to note that the block-minimization formulation of MKL presented in [10, 2] is equivalent to (1).

A straightforward approach toward solving the convex-concave problem in (1) is to transform it into a Semi-definite Programming (SDP) or a Quadratically Constrained Quadratic Programming (QCQP) [5, 2]. However, given their computational complexity, they cannot be applied to large-scale MKL problems. Recently, Semi-infinite Linear Programming (SILP) [12] and Subgradient Descent (SD) [10] have been applied to handle large-scale MKL problems. We summarize them into a unified framework in Algorithm 1. Note that a superscript is used to indicate the index of iteration, a convention that is used throughout this paper. We use $[x]^t$ to denote $x$ to the power of $t$ in the case of ambiguity.

As indicated in Algorithm 1, both methods divide the MKL problem into two cycles: the inner cycle solves a standard SVM problem to update $\alpha$, and the outer cycle updates the kernel weight vector

**Algorithm 1** A general framework for solving MKL
___
1: Initialize $\mathbf{p}^0 = \mathbf{e}/m$ and $i = 0$
2: **repeat**
3:   Solve the dual of SVM with kernel $\mathbf{K} = \sum_{j=1}^m p_j^i \mathbf{K}_j$ and obtain optimal solution $\alpha^i$
4:   Update kernel weights by $\mathbf{p}^{i+1} = \arg\min\{\varphi^i(\mathbf{p}; \alpha) : \mathbf{p} \in \mathcal{P}\}$
5:   Update $i = i + 1$ and calculate stopping criterion $\Delta^i$
6: **until** $\Delta^i \leq \varepsilon$
___

$\mathbf{p}$. They differ in the 4th step in Algorithm 1: the SILP method updates $\mathbf{p}$ by solving a cutting plane model, while the SD method updates $\mathbf{p}$ using the subgradient of the current solution. More specifically, $\varphi^i(\mathbf{p}; \alpha)$ for SILP and SD are defined as follows:

$$\varphi_{SILP}^i(\mathbf{p}; \alpha) = \min_\nu \{\nu : \nu \geq f(\mathbf{p}, \alpha^j), \; j = 0, \dots, i\}, \tag{2}$$

$$\varphi_{SD}^i(\mathbf{p}; \alpha) = \frac{1}{2}\|\mathbf{p} - \mathbf{p}^i\|_2^2 + \gamma_i(\mathbf{p} - \mathbf{p}^i)^\top \nabla_\mathbf{p} f(\mathbf{p}^i, \alpha^i), \tag{3}$$

where $\gamma_i$ is the step size that needs to be decided dynamically (e.g., by a line search). $\nabla_\mathbf{p} f(\mathbf{p}^i, \alpha^i) = -\frac{1}{2}[(\alpha^i \circ \mathbf{y})^\top \mathbf{K}_1 (\alpha^i \circ \mathbf{y}), \dots, (\alpha^i \circ \mathbf{y})^\top \mathbf{K}_m (\alpha^i \circ \mathbf{y})]^\top$ denotes the subgradient of $f(\cdot, \cdot)$ with respect to $\mathbf{p}$ at $(\mathbf{p}^i, \alpha^i)$. Comparing the two methods, we observe

- In SILP, the cutting plane model $\varphi_{SILP}(\mathbf{p})$ utilizes all the $\{\alpha^j\}_{j=1}^i$ obtained in past iterations. In contrast, SD only utilizes $\alpha^i$ of the current solution $\mathbf{p}^i$.
- SILP updates the solution for $\mathbf{p}$ based on the cutting plane model $\varphi_{SILP}(\mathbf{p})$. Since the cutting plane model is usually inaccurate when $\mathbf{p}$ is far away from $\{\mathbf{p}^j\}_{j=1}^i$, the updated solution $\mathbf{p}$ could be significantly off target [3]. In contrast, a regularization term $\|\mathbf{p} - \mathbf{p}^i\|_2^2/2$ is introduced in SD to prevent the new solution being far from the current one, $\mathbf{p}^i$.

The proposed level method combines the strengths of both methods. Similar to SILP, it utilizes the gradient information of all the iterations; similar to SD, a regularization scheme is introduced to prevent the updated solution from being too far from the current solution.

## 3  Extended Level Method for MKL

We first introduce the basic steps of the level method, followed by the extension of the level method to convex-concave problems and its application to MKL.

### 3.1  Introduction to the Level Method

The level method [6] is from the family of bundle methods, which have recently been employed to efficiently solve regularized risk minimization problems [11]. It is an iterative approach designed for optimizing a non-smooth objective function. Let $f(x)$ denote the convex objective function to be minimized over a convex domain $G$. In the $i$th iteration, the level method first constructs a lower bound for $f(x)$ by a cutting plane model, denoted by $g^i(x)$. The optimal solution, denoted by $\hat{x}^i$, that minimizes the cutting plane model $g^i(x)$ is then computed. An upper bound $\overline{f}^i$ and a lower bound $\underline{f}_i$ are computed for the optimal value of the target optimization problem based on $\hat{x}^i$. Next, a level set for the cutting plane model $g^i(x)$ is constructed, denoted by $\mathcal{L}^i = \{x \in G : g^i(x) \leq \lambda \overline{f}^i + (1 - \lambda)\underline{f}^i\}$ where $\lambda \in (0, 1)$ is a tradeoff constant. Finally, a new solution $x^{i+1}$ is computed by projecting $x^i$ onto the level set $\mathcal{L}^i$. It is important to note that the projection step, serving a similar purpose to the regularization term in SD, prevents the new solution $x^{i+1}$ from being too far away from the old one $x^i$. To demonstrate this point, consider a simple example $\min_x\{f(x) = [x]^2 : x \in [-4, 4]\}$. Assume $x^0 = -3$ is the initial solution. The cutting plane model at $x^0$ is $g^0(x) = 9 - 6(x + 3)$. The optimal solution minimizing $g^0(x)$ is $\hat{x}^1 = 4$. If we directly take $\hat{x}^1$ as the new solution, as SILP does, we found it is significantly worse than $x^0$ in terms of $[x]^2$. The level method alleviates this problem by projecting $x^0 = -3$ to the level set $\mathcal{L}^0 = \{x : g^0(x) \leq 0.9[x^0]^2 + 0.1g^0(\hat{x}^1), -4 \leq x \leq 4\}$ where $\lambda = 0.9$. It is easy to verify that

the projection of $x^0$ to $\mathcal{L}^0$ is $x^1 = -2.3$, which significantly reduces the objective function $f(x)$ compared with $x^0$.

## 3.2 Extension of the Level Method to MKL

We now extend the level method, which was originally designed for optimizing non-smooth functions, to convex-concave optimization. First, since $f(\mathbf{p}, \alpha)$ is convex in $\mathbf{p}$ and concave in $\alpha$, according to van Neuman Lemma, for any optimal solution $(\mathbf{p}^*, \alpha^*)$ we have

$$f(\mathbf{p}, \alpha^*) = \max_{\alpha \in \mathcal{Q}} f(\mathbf{p}, \alpha) \geq f(\mathbf{p}^*, \alpha^*) \geq f(\mathbf{p}^*, \alpha) = \min_{\mathbf{p} \in \mathcal{P}} f(\mathbf{p}, \alpha). \tag{4}$$

This observation motivates us to design an MKL algorithm which iteratively updates both the lower and the upper bounds for $f(\mathbf{p}, \alpha)$ in order to find the saddle point. To apply the level method, we first construct the cutting plane model. Let $\{\mathbf{p}^j\}_{j=1}^i$ denote the solutions for $\mathbf{p}$ obtained in the last $i$ iterations. Let $\alpha^j = \arg\max_{\alpha \in \mathcal{Q}} f(\mathbf{p}^j, \alpha)$ denote the optimal solution that maximizes $f(\mathbf{p}^j, \alpha)$. We construct a cutting plane model $g^i(\mathbf{p})$ as follows:

$$g^i(\mathbf{p}) = \max_{1 \leq j \leq i} f(\mathbf{p}, \alpha^j). \tag{5}$$

We have the following proposition for the cutting plane model $g^i(x)$

**Proposition 1.** *For any $\mathbf{p} \in \mathcal{P}$, we have (a) $g^{i+1}(\mathbf{p}) \geq g^i(\mathbf{p})$, and (b) $g^i(\mathbf{p}) \leq \max_{\alpha \in \mathcal{Q}} f(\mathbf{p}, \alpha)$.*

Next, we construct both the lower and the upper bounds for the optimal value $f(\mathbf{p}^*, \alpha^*)$. We define two quantities $\underline{f}^i$ and $\overline{f}^i$ as follows:

$$\underline{f}^i = \min_{\mathbf{p} \in \mathcal{P}} g^i(\mathbf{p}) \quad and \quad \overline{f}^i = \min_{1 \leq j \leq i} f(\mathbf{p}^j, \alpha^j). \tag{6}$$

The following theorem shows that $\{\underline{f}^j\}_{j=1}^i$ and $\{\overline{f}^j\}_{j=1}^i$ provide a series of increasingly tight bounds for $f(\mathbf{p}^*, \alpha^*)$.

**Theorem 1.** *We have the following properties for $\{\underline{f}^j\}_{j=1}^i$ and $\{\overline{f}^j\}_{j=1}^i$: (a) $\underline{f}^i \leq f(\mathbf{p}^*, \alpha^*) \leq \overline{f}^i$, (b) $\overline{f}^1 \geq \overline{f}^2 \geq \ldots \geq \overline{f}^i$, and (c) $\underline{f}^1 \leq \underline{f}^2 \leq \ldots \leq \underline{f}^i$.*

*Proof.* First, since $g^i(\mathbf{p}) \leq \max_{\alpha \in \mathcal{Q}} f(\mathbf{p}, \alpha)$ for any $\mathbf{p} \in \mathcal{P}$, we have

$$\underline{f}^i = \min_{\mathbf{p} \in \mathcal{P}} g^i(\mathbf{p}) \leq \min_{\mathbf{p} \in \mathcal{P}} \max_{\alpha \in \mathcal{Q}} f(\mathbf{p}, \alpha).$$

Second, since $f(\mathbf{p}^j, \alpha^j) = \max_{\alpha \in Q} f(\mathbf{p}^j, \alpha)$, we have

$$\overline{f}^i = \min_{1 \leq j \leq i} f(\mathbf{p}^j, \alpha^j) = \min_{\mathbf{p} \in \{\mathbf{p}_1, \ldots, \mathbf{p}_i\}} \max_{\alpha \in \mathcal{Q}} f(\mathbf{p}, \alpha) \geq \min_{\mathbf{p} \in \mathcal{P}} \max_{\alpha \in \mathcal{Q}} f(\mathbf{p}, \alpha) = f(\mathbf{p}^*, \alpha^*).$$

Combining the above results, we have (a) in the theorem. It is easy to verify (b) and (c). $\square$

We furthermore define the gap $\Delta^i$ as

$$\Delta^i = \overline{f}^i - \underline{f}^i.$$

The following corollary indicates that the gap $\Delta^i$ can be used to measure the sub-optimality for solution $\mathbf{p}^i$ and $\alpha^i$.

**Corollary 2.** *(a) $\Delta^j \geq 0, j = 1, \ldots, i$, (b) $\Delta^1 \geq \Delta^2 \geq \ldots \geq \Delta^i$, (c) $|f(\mathbf{p}^j, \alpha^j) - f(\mathbf{p}^*, \alpha^*)| \leq \Delta^i$*

It is easy to verify these three properties of $\Delta^i$ in the above corollary using the results of Theorem 1.

In the third step, we construct the level set $\mathcal{L}^i$ using the estimated bounds $\overline{f}^i$ and $\underline{f}^i$ as follows:

$$\mathcal{L}^i = \{\mathbf{p} \in \mathcal{P} : g^i(\mathbf{p}) \leq \ell^i = \lambda \overline{f}^i + (1 - \lambda) \underline{f}^i\}, \tag{7}$$

where $\lambda \in (0,1)$ is a predefined constant. The new solution, denoted by $\mathbf{p}^{i+1}$, is computed as the projection of $\mathbf{p}^i$ onto the level set $\mathcal{L}^i$, which is equivalent to solving the following optimization problem:

$$\mathbf{p}^{i+1} = \arg\min_{\mathbf{p}} \left\{ \|\mathbf{p} - \mathbf{p}^i\|_2^2 : \mathbf{p} \in \mathcal{P}, f(\mathbf{p}, \alpha^j) \leq \ell^i, j = 1, \ldots, i \right\}. \tag{8}$$

Although the projection is regarded as a quadratic programming problem, it can often be solved efficiently because its solution is likely to be the projection onto one of the hyperplanes of polyhedron $\mathcal{L}^i$. In other words, only very few linear constraints of $\mathcal{L}$ are active; most of them are inactive. This sparse nature usually leads to significant speedup of QP, similar to the solver of SVM. As we argue in the last subsection, by means of the projection, we on the one hand ensure $\mathbf{p}^{i+1}$ is not very far away from $\mathbf{p}^i$, and on the other hand ensure significant progress is made in terms of $g^i(\mathbf{p})$ when the solution is updated from $\mathbf{p}^i$ to $\mathbf{p}^{i+1}$. Note that the projection step in the level method saves the effort of searching for the optimal step size in SD, which is computationally expensive as will be revealed later. We summarize the steps of the extended level method in Algorithm 2.

---

**Algorithm 2** The Level Method for Multiple Kernel Learning

---
1: Initialize $\mathbf{p}^0 = \mathbf{e}/m$ and $i = 0$
2: **repeat**
3:    Solve the dual problem of SVM with $\mathbf{K} = \sum_{j=1}^m p_j^i \mathbf{K}_j$ to obtain the optimal solution $\alpha^i$
4:    Construct the cutting plane model $g^i(\mathbf{p})$ in (5)
5:    Calculate the lower bound $\underline{f}^i$ and the upper bound $\overline{f}^i$ in (6), and the gap $\Delta^i$ in (3.2)
6:    Compute the projection of $\mathbf{p}^i$ onto the level set $\mathcal{L}^i$ by solving the optimization problem in (8)
7:    Update $i = i + 1$
8: **until** $\Delta^i \leq \varepsilon$

---

Finally, we discuss the convergence behavior of the level method. In general, convergence is guaranteed because the gap $\Delta^i$, which bounds the absolute difference between $f(\mathbf{p}^*, \alpha^*)$ and $f(\mathbf{p}^i, \alpha^i)$, monotonically decreases through iterations. The following theorem shows the convergence rate of the level method when applied to multiple kernel learning.

**Theorem 3.** *To obtain a solution* $\mathbf{p}$ *that satisfies the stopping criterion, i.e.,* $|\max_{\alpha \in \mathcal{Q}} f(\mathbf{p}, \alpha) - f(\mathbf{p}^*, \alpha^*)| \leq \varepsilon$, *the maximum number of iterations* $N$ *that the level method requires is bounded as follows* $N \leq \frac{2c(\lambda)L^2}{\varepsilon^2}$, *where* $c(\lambda) = \frac{1}{(1-\lambda)^2 \lambda(2-\lambda)}$ *and* $L = \frac{1}{2}\sqrt{m}nC^2 \max_{1 \leq i \leq m} \Lambda_{\max}(\mathbf{K}_i)$. *The operator* $\Lambda_{\max}(M)$ *computes the maximum eigenvalue of matrix* $M$.

Due to space limitation, the proof of Theorem 3 can be found in the long version of this paper. Theorem 3 tells us that the convergence rate of the level method is $\mathcal{O}(1/\varepsilon^2)$. It is important to note that according to Information Based Complexity (IBC) theory, given a function family $\mathcal{F}(L)$ with a fixed Lipschitz constant $L$, $\mathcal{O}(1/\varepsilon^2)$ is almost the optimal convergence rate that can be achieved for any optimization method based on the black box first order oracle. In other words, no matter which optimization method is used, there always exists an function $f(\cdot) \in \mathcal{F}(L)$ such that the convergence rate is $\mathcal{O}(1/\varepsilon^2)$ as long as the optimization method is based on a black box first order oracle. More details can be found in [8, 6].

# 4 Experiments

We conduct experiments to evaluate the efficiency of the proposed algorithm for MKL in constrast with SILP and SD, the two state-of-the-art algorithms for MKL.

## 4.1 Experimental Setup

We follow the settings in [10] to construct the base kernel matrices, i.e.,

- Gaussian kernels with 10 different widths ($\{2^{-3}, 2^{-2}, \ldots, 2^6\}$) on all features and on each single feature
- Polynomial kernels of degree 1 to 3 on all features and on each single feature.

Table 1: The performance comparison of three MKL algorithms. Here $n$ and $m$ denote the size of training samples and the number of kernels, respectively.

| | SD | SILP | Level | SD | SILP | Level |
|---|---|---|---|---|---|---|
| | Iono | $n=175$ | $m=442$ | Breast | $n=342$ | $m=117$ |
| Time(s) | 33.5 $\pm$11.6 | 1161.0 $\pm$344.2 | 7.1 $\pm$4.3 | 47.4 $\pm$8.9 | 54.2 $\pm$9.4 | 4.6 $\pm$1.0 |
| Accuracy (%) | 92.1 $\pm$2.0 | 92.0 $\pm$1.9 | 92.1$\pm$1.9 | 96.6 $\pm$0.9 | 96.6 $\pm$0.8 | 96.6$\pm$0.8 |
| #Kernel | 26.9 $\pm$4.0 | 24.4 $\pm$3.4 | 25.4$\pm$3.9 | 13.1 $\pm$1.7 | 10.6 $\pm$1.1 | 13.3$\pm$1.5 |
| | Pima | $n=384$ | $m=117$ | Sonar | $n=104$ | $m=793$ |
| Time(s) | 39.4 $\pm$8.8 | 62.0 $\pm$15.2 | 9.1 $\pm$1.6 | 60.1 $\pm$29.6 | 1964.3$\pm$68.4 | 24.9$\pm$10.6 |
| Accuracy (%) | 76.9 $\pm$1.9 | 76.9 $\pm$2.1 | 76.9$\pm$2.1 | 79.1 $\pm$4.5 | 79.3 $\pm$4.2 | 79.0$\pm$4.7 |
| #Kernel | 16.6 $\pm$2.2 | 12.0 $\pm$1.8 | 17.6$\pm$2.6 | 39.8 $\pm$3.9 | 34.2 $\pm$2.6 | 38.6$\pm$4.1 |
| | Wpbc | $n=198$ | $m=442$ | Heart | $n=135$ | $m=182$ |
| Time(s) | 7.8 $\pm$2.4 | 142.0 $\pm$122.3 | 5.3 $\pm$1.3 | 4.7 $\pm$2.8 | 79.2 $\pm$38.1 | 2.1 $\pm$0.4 |
| Accuracy (%) | 77.0 $\pm$2.9 | 76.9 $\pm$2.8 | 76.9$\pm$2.9 | 82.2 $\pm$2.2 | 82.2 $\pm$2.0 | 82.2$\pm$2.1 |
| #Kernel | 19.5 $\pm$2.8 | 17.2 $\pm$2.2 | 20.3$\pm$2.6 | 17.5 $\pm$1.8 | 15.2 $\pm$1.5 | 18.6$\pm$1.9 |
| | Vote | $n=218$ | $m=205$ | Wdbc | $n=285$ | $m=403$ |
| Time(s) | 23.7 $\pm$9.7 | 26.3 $\pm$12.4 | 4.1 $\pm$1.3 | 122.9$\pm$38.2 | 146.3 $\pm$48.3 | 15.5$\pm$7.5 |
| Accuracy (%) | 95.7 $\pm$1.0 | 95.7 $\pm$1.0 | 95.7$\pm$1.0 | 96.7 $\pm$0.8 | 96.5 $\pm$0.9 | 96.7$\pm$0.8 |
| #Kernel | 14.0 $\pm$3.6 | 10.6 $\pm$2.6 | 13.8$\pm$2.6 | 16.6 $\pm$3.2 | 12.9 $\pm$2.3 | 15.6$\pm$3.0 |

Each base kernel matrix is normalized to unit trace. The experiments are conducted on a PC with 3.2GHz CPU and 2GB memory. According to the above scheme of constructing base kernel matrices, we select a batch of UCI data sets, with the cardinality and dimension allowed by the memory limit of the PC, from the UCI repository for evaluation. We repeat all the algorithms 20 times for each data set. In each run, 50% of the examples are randomly selected as the training data and the remaining data are used for testing. The training data are normalized to have zero mean and unit variance, and the test data are then normalized using the mean and variance of the training data. The regularization parameter $C$ in SVM is set to $100$ as our focus is to evaluate the computational time, as justified in [10]. For a fair comparison among the MKL algorithms, we adopt the same stopping criterion for all three algorithms under comparison: we adopt the duality gap criterion used in [10], i.e., $\max_{1 \leq i \leq m} (\alpha \circ \mathbf{y})^\top \mathbf{K}_i (\alpha \circ \mathbf{y}) - (\alpha \circ \mathbf{y})^\top \left( \sum_{j=1}^m p_j \mathbf{K}_j \right) (\alpha \circ \mathbf{y})$, and stop the algorithm when the criterion is less than $0.01$ or the number of iterations larger than $500$. We empirically initialize the parameter $\lambda$ to $0.9$ and increase it to $0.99$ when the ratio $\Delta_i / \ell_i$ is less than $0.01$ for all experiments, since a larger $\lambda$ accelerates the projection when the solution is close to the optimal one. We use the SimpleMKL toolbox [10] to implement the SILP and SD methods. The linear programming in the SILP method and the auxiliary subproblems in the level method are solved using a general optimization toolbox MOSEK (http://www.mosek.com). The toolbox for the level method can be downloaded from http://www.cse.cuhk.edu.hk/~zlxu/toolbox/level_mkl.html.

## 4.2 Experimental Results

We report the following performance measures: prediction accuracy, training time, and the averaged number of kernels selected. From Table 1, we observe that all algorithms achieve almost the same prediction accuracy under the same stopping criterion. This is not surprising because all algorithms are essentially trying to solve the same optimization problem. Regarding the computational efficiency, we observe that the time cost of the SILP approach is the highest among all the three MKL algorithms. For datasets "Iono" and "Sonar", the SILP method consumes more than 30 times the computational cycles of the other two methods for MKL. We also observe that the level method is the most efficient among three methods in comparison. To obtain a better picture of the computational efficiency of the proposed level method, we compute the time-saving ratio, as shown in Table 2. We observe that the level method saves $91.9\%$ of computational time on average when compared with the SILP method, and $70.3\%$ of computational time when compared with the SD method.

In order to see more details of each optimization algorithm, we plot the logarithm values of the MKL objective function to base 10 against time in Figure 1. Due to space limitation, we randomly choose only three datasets, "Iono", "Breast", and "Pima", as examples. It is interesting to find that the level method converges overwhelmingly faster than the other two methods. The efficiency of the level method arises from two aspects: (a) the cutting plane model utilizes the computational results of all iterations and therefore boosts the search efficiency, and (b) the projection to the level sets ensures the stability of the new solution. A detailed analysis of the SD method reveals that a large

number of function evaluations are consumed in order to compute the optimal stepsize via a line search. Note that in convex-concave optimization, every function evaluation in the line search of SD requires solving an SVM problem. As an example, we found that for dataset "Iono", although SD and the level method require similar numbers of iterations, SD calls the SVM solver 1231 times on average, while the level method only calls it 47 times. For the SILP method, the high computational cost is mainly due to the oscillation of solutions. This instability leads to very slow convergence when the solution is close to the optimal one, as indicated by the long tail of SILP in Figure 1. The instability of SILP is further confirmed by the examination of kernel weights, as shown below.

To understand the evolution of kernel weights (i.e., $\mathbf{p}$), we plot the evolution curves of the five largest kernel weights for datasets "Iono", "Breast", and "Pima" in Figure 2. We observe that the values of $\mathbf{p}$ computed by the SILP method are the most unstable due to oscillation of the solutions to the cutting plane models. Although the unstable-solution problem is to some degree improved by the SD method, we still clearly observe that $\mathbf{p}$ fluctuates significantly through iterations. In contrast, for the proposed level method, the values of $\mathbf{p}$ change smoothly through iterations. We believe that the stability of the level method is mainly due to the accurate estimation of bounds as well as the regularization of the projection to the level sets. This observation also sheds light on why the level method can be more efficient than the SILP and the SD methods.

Table 2: Time-saving ratio of the level method over the SILP and the SD method

|  | Iono | Breast | Pima | Sonar | Wpbc | Heart | Vote | Wdbc | Average |
|---|---|---|---|---|---|---|---|---|---|
| Level/SD (%) | 78.9 | 90.4 | 77.0 | 58.7 | 32.5 | 54.7 | 82.8 | 87.4 | 70.3 |
| Level/SILP (%) | 99.4 | 91.6 | 85.4 | 98.7 | 88.7 | 97.3 | 84.5 | 89.4 | 91.9 |

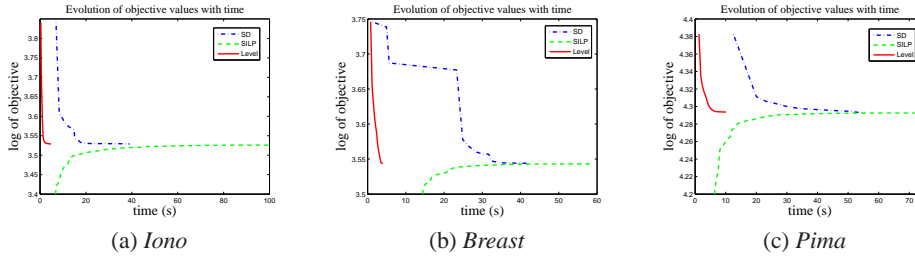

(a) *Iono*      (b) *Breast*      (c) *Pima*

Figure 1: Evolution of objective values over time (seconds) for datasets "Iono", "Breast", and "Pima". The objective values are plotted on a logarithm scale (base 10) for better comparison. Only parts of the evolution curves are plotted for SILP due to their long tails.

## 5 Conclusion and Future Work

In this paper, we propose an extended level method to efficiently solve the multiple kernel learning problem. In particular, the level method overcomes the drawbacks of both the SILP method and the SD method for MKL. Unlike the SD method that only utilizes the gradient information of the current solution, the level method utilizes the gradients of all the solutions that are obtained in past iterations; meanwhile, unlike the SILP method that updates the solution only based on the cutting plane model, the level method introduces a projection step to regularize the updated solution. It is the employment of the projection step that guarantees finding an updated solution that, on the one hand, is close to the existing one, and one the other hand, significantly reduces the objective function. Our experimental results have shown that the level method is able to greatly reduce the computational time of MKL over both the SD method and the SILP method. For future work, we plan to find a scheme to adaptively set the value of $\lambda$ in the level method and apply the level method to other tasks, such as one-class classification, multi-class classification, and regression.

## Acknowledgement

The work was supported by the National Science Foundation (IIS-0643494), National Institute of Health (1R01GM079688-01) and Research Grants Council of Hong Kong (CUHK4150/07E and CUHK4125/07).

## References

[1] F. R. Bach. Consistency of the group Lasso and multiple kernel learning. *Journal of Machine Learning Research*, 9:1179–1225, 2008.

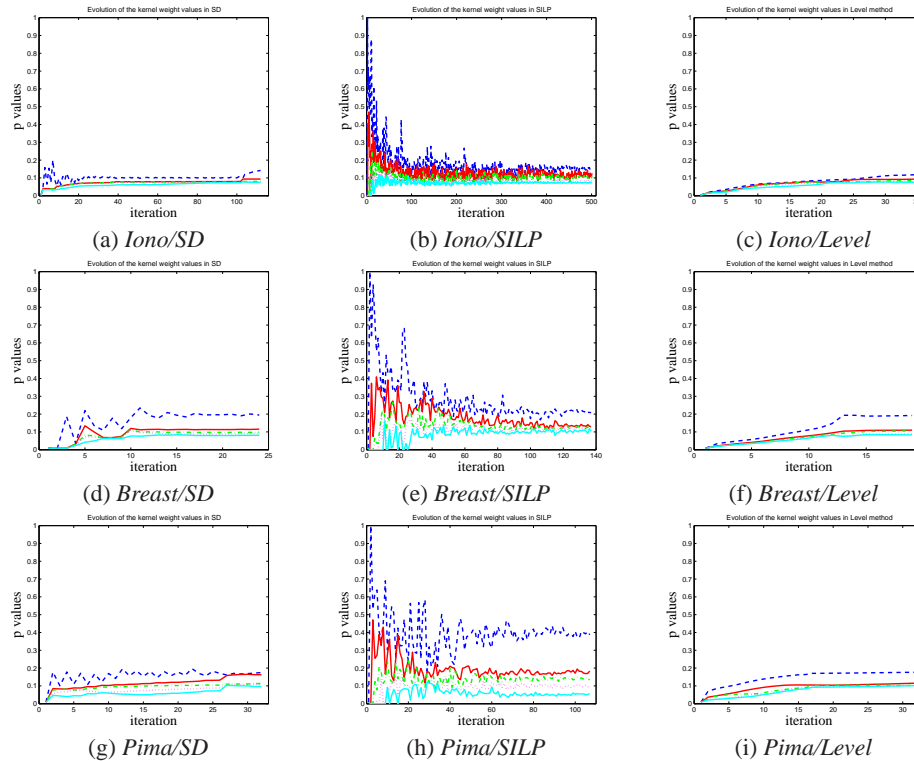

Figure 2: The evolution curves of the five largest kernel weights for datasets "Iono", "Breast" and "Pima" computed by the three MKL algorithms

[2] F. R. Bach, G. R. G. Lanckriet, and M. I. Jordan. Multiple kernel learning, conic duality, and the SMO algorithm. In *ICML*, 2004.

[3] J. Bonnans, J. Gilbert, C. Lemaréchal, and C. Sagastizábal. *Numerical Optimization, Theoretical and Practical Aspects*. Springer-Verlag, Berlin, 2nd ed., 2006.

[4] N. Cristianini, J. Shawe-Taylor, A. Elisseeff, and J. S. Kandola. On kernel-target alignment. In *NIPS 13*, pages 367–373, 2001.

[5] G. R. G. Lanckriet, N. Cristianini, P. Bartlett, L. E. Ghaoui, and M. I. Jordan. Learning the kernel matrix with semidefinite programming. *Journal of Machine Learning Research*, 5, 2004.

[6] C. Lemaréchal, A. Nemirovski, and Y. Nesterov. New variants of bundle methods. *Mathematical Programming*, 69(1), 1995.

[7] C. A. Micchelli and M. Pontil. Learning the kernel function via regularization. *Journal of Machine Learning Research*, 6, 2005.

[8] A. Nemirovski and D. Yudin. *Problem Complexity and Method Efficiency in Optimization*. John Wiley and Sons Ltd, 1983.

[9] C. S. Ong, A. J. Smola, and R. C. Williamson. Learning the kernel with hyperkernels. *Journal of Machine Learning Research*, 6, 2005.

[10] A. Rakotomamonjy, F. R. Bach, S. Canu, and Y. Grandvalet. SimpleMKL. Technical Report HAL-00218338, INRIA, 2008.

[11] A. Smola, S. V. N. Vishwanathan, and Q. Le. Bundle methods for machine learning. In *NIPS 20*, pages 1377–1384, 2007.

[12] S. Sonnenburg, G. Rätsch, C. Schäfer, and B. Schölkopf. Large scale multiple kernel learning. *Journal of Machine Learning Research*, 7, 2006.

[13] J. Ye, J. Chen, and S. Ji. Discriminant kernel and regularization parameter learning via semidefinite programming. In *ICML*, 2007.

[14] A. Zien and C. S. Ong. Multiclass multiple kernel learning. In *ICML*, 2007.
